# Multi-Grid Methods for Reinforcement Learning in Controlled Diffusion Processes

**Stephan Pareigis**
stp@numerik.uni-kiel.de
Lehrstuhl Praktische Mathematik
Christian-Albrechts-Universität Kiel
Kiel, Germany

## Abstract

Reinforcement learning methods for discrete and semi-Markov decision problems such as Real-Time Dynamic Programming can be generalized for Controlled Diffusion Processes. The optimal control problem reduces to a boundary value problem for a fully nonlinear second-order elliptic differential equation of Hamilton-Jacobi-Bellman (HJB-) type. Numerical analysis provides multi-grid methods for this kind of equation. In the case of Learning Control, however, the systems of equations on the various grid-levels are obtained using observed information (transitions and local cost). To ensure consistency, special attention needs to be directed toward the type of time and space discretization during the observation. An algorithm for multi-grid observation is proposed. The multi-grid algorithm is demonstrated on a simple queuing problem.

## 1  Introduction

Controlled Diffusion Processes (CDP) are the analogy to Markov Decision Problems in continuous state space and continuous time. A CDP can always be discretized in state space and time and thus reduced to a Markov Decision Problem. Algorithms like $Q$-learning and RTDP as described in [1] can then be applied to produce controls or optimal value functions for a fixed discretization.

Problems arise when the discretization needs to be refined, or when multi-grid information needs to be extracted to accelerate the algorithm. The relation of time to state space discretization parameters is crucial in both cases. Therefore

a mathematical model of the discretized process is introduced, which reflects the properties of the converged empirical process. In this model, transition probabilities of the discrete process can be expressed in terms of the transition probabilities of the continuous process. Recent results in numerical methods for stochastic control problems in continuous time can be applied to give assumptions that guarantee a local consistency condition which is needed for convergence. The same assumptions allow application of multi-grid methods.

In section 2 Controlled Diffusion Processes are introduced. A model for the discretized process is suggested in section 3 and the main theorem is stated. Section 4 presents an algorithm for multi-grid observation according to the results in the preceding section. Section 5 shows an application of multi-grid techniques for observed processes.

## 2   Controlled Diffusion Processes

Consider a Controlled Diffusion Process (CDP) $\xi(t)$ in some bounded domain $\Omega \subset \mathbb{R}^n$ fulfilling the diffusion equation

$$d\xi(t) = b(\xi(t), u(t))dt + \sigma(\xi(t))dw. \tag{1}$$

The control $u(t)$ takes values in some finite set $U$. The immediate reinforcement (cost) for state $\xi(t)$ and control $u(t)$ is

$$r(t) = r(\xi(t), u(t)). \tag{2}$$

The control objective is to find a feedback control law

$$u(t) = \mathbf{u}(\xi(t)), \tag{3}$$

that minimizes the total discounted cost

$$J(x, u) = \mathbb{E}_x^u \int_0^\infty e^{-\beta t} r(\xi(t), u(t))dt, \tag{4}$$

where $\mathbb{E}_x^u$ is the expectation starting in $x \in \Omega$ and applying the control law $u(.)$. $\beta > 0$ is the discount.

The transition probabilities of the CDP are given for any initial state $x \in \Omega$ and subset $A \subset \Omega$ by the stochastic kernels

$$P_t^u(x, A) := \text{prob}\{\xi(t) \in A \mid \xi(0) = x, u\}. \tag{5}$$

It is known that the kernels have the properties

$$\int_\Omega (y - x) P_t^u(x, dy) = t \cdot b(x, u) + o(t) \tag{6}$$

$$\int_\Omega (y - x)(y - x)^T P_t^u(x, dy) = t \cdot \sigma(x)\sigma(x)^T + o(t). \tag{7}$$

For the optimal control it is sufficient to calculate the optimal value function $V : \Omega \to \mathbb{R}$

$$V(x) := \inf_{u(.)} J(x, u). \tag{8}$$

Under appropriate smoothness assumptions $V$ is a solution of the Hamilton-Jacobi-Bellman (HJB-) equation

$$\min_{\alpha \in U} \{ \mathcal{L}^\alpha V(x) - \beta V(x) + r(x, \alpha) \} = 0, \quad x \in \Omega. \tag{9}$$

Let $a(x) = \sigma(x)\sigma(x)^T$ be the diffusion matrix, then $\mathcal{L}^\alpha$, $\alpha \in U$ is defined as the elliptic differential operator

$$\mathcal{L}^\alpha := \sum_{i,j=1}^{n} a_{ij}(x)\partial_{x_i}\partial_{x_j} + \sum_{i=1}^{n} b_i(x, \alpha)\partial_{x_i}. \tag{10}$$

# 3   A Model for Observed CDP's

Let $\Omega_{h_i}$ be the centers of cells of a cell-centered grid on $\Omega$ with cell sizes $h_0$, $h_1 = h_0/2$, $h_2 = h_1/2$, .... For any $x \in \Omega_{h_i}$ we shall denote by $A(x)$ the cell of $x$. Let $\Delta t > 0$ be a parameter for the time discretization.

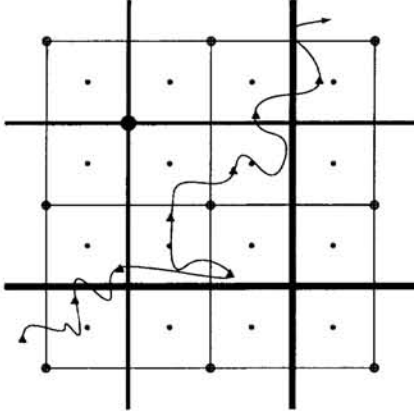

Figure 1: The picture depicts three cell-centered grid levels and the trajectory of a diffusion process. The approximating value function is represented locally constant on each cell. The triangles on the path denote the position of the diffusion at sample times $0, \Delta t, 2\Delta t, 3\Delta t, \ldots$. Transitions between respective cells are then counted in matrices $Q_i^a$, for each control $a$ and grid $i$.

By counting the transitions between cells and calculating the empirical probabilities as defined in (20) we obtain empirical processes on every grid. By the law of great numbers the empirical processes will converge towards observed CDPs as subsequently defined.

**Definition 1** *An observed process $\xi_{h_i, \Delta t_i}(t)$ is a Controlled Markov Chain (i.e. discrete state-space and discrete time) on $\Omega_{h_i}$ and interpolation time $\Delta t_i$ with the transition probabilities*

$$
\begin{aligned}
p_{h_i, \Delta t_i}^u(x, y) &:= \quad prob\{\xi(\Delta t_i) \in A(y) | \xi(0) \in A(x), u\} \\
&= \quad \frac{1}{h_i^n} \int_{A(x)} P_{\Delta t_i}^u(z, A(y)) dz,
\end{aligned} \tag{11}
$$

*where $x, y \in \Omega_{h_i}$ and $\xi(t)$ is a solution of (1). Also define the observed reinforcement $\rho$ as*

$$\rho_{h_i, \Delta t_i}(x, u) := \frac{1}{h_i^n} \int_{A(x)} \mathbb{E}_z^u \int_0^{\Delta t_i} r(\xi(\tau), u(\tau)) d\tau dz. \tag{12}$$

On every grid $\Omega_{h_i}$ the respective process $\xi_{h_i,\Delta t_i}$ has its own value function $V_{h_i,\Delta t_i}$. By theorem 10.4.1. in Kushner, Dupuis ([5], 1992) it holds, that

$$V_{h_i,\Delta t_i}(x) \to V(x) \text{ for all } x \in \Omega, \tag{13}$$

if the following local consistency conditions hold.

**Definition 2** *Let $\Delta\xi_{h,\Delta t} = \xi_{h,\Delta t}(\Delta t) - \xi_{h,\Delta t}(0)$. $\xi_{h,\Delta t}$ is called locally consistent to a solution $\xi(.)$ of (1), iff*

$$\mathbb{E}_x^u \Delta\xi_{h,\Delta t} = b(x,a)\Delta t + o(\Delta t) \tag{14}$$

$$\mathbb{E}_x^u[\Delta\xi_{h,\Delta t} - \mathbb{E}_x^u\Delta\xi_{h,\Delta t}][\Delta\xi_{h,\Delta t} - \mathbb{E}_x^u\Delta\xi_{h,\Delta t}]^T = a(x)\Delta t + o(\Delta t) \tag{15}$$

$$\sup_n |\Delta\xi_{h,\Delta t}(n\Delta t)| \to 0 \text{ as } h \to 0. \tag{16}$$

To verify these conditions for the observed CDP, the expectation and variance can be calculated. For the expectation we get

$$
\begin{aligned}
\mathbb{E}_x^u \Delta\xi_{h_i,\Delta t_i} &= \sum_{y \in \Omega_{h_i}} p_{h_i,\Delta t_i}^u(x,y)(y-x) \\
&= \frac{1}{h_i^n} \sum_{y \in \Omega_{h_i}} \int_{A(x)} (y-x)P_{\Delta t_i}^u(z, A(y))dz. \tag{17}
\end{aligned}
$$

Recalling properties (6) and (7) and doing a similar calculation for the variance we obtain the following theorem.

**Theorem 3** *For observed CDPs $\xi_{h_i,\Delta t_i}$ let $h_i$ and $\Delta t_i$ be such that*

$$h_i/\Delta t_i \to 0 \text{ as } \Delta t_i \to 0. \tag{18}$$

*Furthermore, $\xi_{h_i,\Delta t_i}$ shall be truncated at some radius $R$, such that $R \to 0$ for $h_i \to 0$ and expectation and variance of the truncated process differ only in the order $o(\Delta t)$ from expectation and variance of $\xi_{h_i,\Delta t_i}$. Then the observed processes $\xi_{h_i,\Delta t_i}$ truncated at $R$ are locally consistent to the diffusion process $\xi(.)$ and therefore the value functions $V_{h_i,\Delta t_i}$ converge to the value function $V$.*

## 4    Identification by Multi-Grid Observation

The condition in Theorem 3 provides information as how to choose parameters in the algorithm with empirical data. Choose discretization values $h_0$, $\Delta t_0$ for the coarsest grid $\Omega_0$. $\Delta t_0$ should typically be of order $\|b\|_{sup}/h_0$. Then choose for the finer grids

| grid | 0 | 1 | 2 | 3 | 4 | 5 | ... | |
|---|---|---|---|---|---|---|---|---|
| space | $h_0$ | $\frac{h_0}{2}$ | $\frac{h_0}{4}$ | $\frac{h_0}{8}$ | $\frac{h_0}{16}$ | $\frac{h_0}{32}$ | ... | |
| time | $\Delta t_0$ | $\frac{\Delta t_0}{2}$ | $\frac{\Delta t_0}{2}$ | $\frac{\Delta t_0}{4}$ | $\frac{\Delta t_0}{4}$ | $\frac{\Delta t_0}{8}$ | ... | (19) |

The sequences verify the assumption (18). We may now formulate the algorithm for Multi-Grid Observation of the CDP $\xi(.)$. Note that only observation is being carried out. The actual calculation of the value function may be done separately as described in the next section. The choice of the control is assumed to be done

by a separate controller. Let $\Omega_k$ be the finest grid, i.e. $\Delta t_k$ and $h_k$ the finest discretizations. Let $U_l = U^{\Delta t_l / \Delta t_k} = U \times \ldots \times U$, $\Delta t_l / \Delta t_k$ times. $Q_l^{a_l}$ is a $|\Omega_l| \times |\Omega_l|$-matrix ($a_l \in U_l$), containing the number of transitions between cells in $\Omega_l$. $R_l^{a_l}$ is a $|\Omega_l|$-vector containing the empirical cost for every cell in $\Omega_l$. The immediate cost is given by the system as $r_l = \int_0^{\Delta t_l} e^{-\beta t} r(\xi(t), a_l) dt$. $T$ denotes current time.

---

0. **Initialize** $\Omega_l, Q_l^{a_l}, R_l^{a_l}$ for all $a_l \in U_l$, $l = 0, \ldots, k$
1. **repeat** {
2.    **choose** $a = a(T) \in U$ and apply $a$ constantly on $[T; T + \Delta t_k]$
3.    $T := T + \Delta t_k$
4.    **for** $l = 0$ to $k$ **do** {
5.      **determine cell** $x_l \in \Omega_l$ *with* $\xi(T - \Delta t_l) \in A(x_l)$
6.      **determine cell** $y_l \in \Omega_l$ *with* $\xi(T) \in A(y_l)$
7.      **if** $\|x_k - y_k\| \geq R$ (truncation radius) **then goto 2. else**
8.      $a_l := (a(T - \Delta t_l), a(T + \Delta t_k - \Delta t_l), \ldots, a(T - \Delta t_k))$
9.      **receive immediate cost** $r_l$
10.     $Q_l^{a_l}(x_l, y_l) := Q_l^{a_l}(x_l, y_l) + 1$
11.     $R_l^{a_l}(x_l) := \left( r_l + R_l^{a_l}(x_l) \cdot \sum_{z \in \Omega_l} Q_l^{a_l}(x_l, z) \right) / \left( 1 + \sum_{z \in \Omega_l} Q_l^{a_l}(x_l, z) \right)$
     } (for-do)
   } (repeat)

---

Before applying a multi-grid algorithm for the calculation of the value function on the basis of the observations, one should make sure that every box has at least some data for every control. Especially in the early stages of learning only the two coarsest grids $\Omega_0, \Omega_1$ could be used for computation of the optimal value function and finer grids may be added (possibly locally) as learning evolves.

## 5   Application of Multi-Grid Techniques

The identification algorithm produces matrices $Q_l^{a_l}$ containing the number of transitions between boxes in $\Omega_l$. We will calculate from the matrices $Q$ the transition matrices $P$ by the formula

$$P_l^{a_l}(x, y) = Q_l^{a_l}(x, y) / \left( \sum_{z \in \Omega_l} Q_l^{a_l}(x, z) \right), \quad x, y \in \Omega_l,\ a_l \in U_l,\ l = 0, \ldots, k. \quad (20)$$

Now we define matrices $A$ and right hand sides $f$ as

$$A_l^{a_l} := (\beta_l P_l^{a_l} - I) / \Delta t_l \quad f_l^{a_l} := R_l^{a_l} / \Delta t_l, \quad (21)$$

where $\beta_l = e^{-\beta \Delta t_l}$. The discrete Bellman equation takes the following form

$$\min_{a_l \in U_l} \{ A_l^{a_l} V_l(x) - f_l^{a_l}(x) \} = 0. \quad (22)$$

The problem is now in a form to which the multi-grid method due to Hoppe, Bloß ([2], 1989) can be applied. For prolongation and restriction we choose bilinear interpolation and full weighted restriction for cell-centered grids. We point out, that for any cell $x \in \Omega_l$ only those neighboring cells shall be used for prolongation and restriction for which the minimum in (22) is attained for the same control as the minimizing control in $x$ (see [2], 1989 and [3], 1996 for details). On every grid

$\Omega_l$ the defect in equation (22) is calculated and used for a correction on grid $\Omega_{l-1}$. As a smoother nonlinear Gauss-Seidel iteration applied to (22) is used.

Our approach differs from the algorithm in Hoppe, Bloß ([2], 1989) in the special form of the matrices $A_l^{a_l}$ in equation (22). The stars are generally larger than nine-point, in fact the stars grow with decreasing $h$ although the matrices remain sparse. Also, when working with empirical information the relationship between the matrices $A_l^{a_l}$ on the various grids is based on observation of a process, which implies that coarse grid corrections do not always correct the equation of the finest grid (especially in the early stages of learning). However, using the observed transition matrices $A_l^{a_l}$ on the coarse grids saves the computing time which would otherwise be needed to calculate these matrices by the Galerkin product (see Hackbusch [4], 1985).

## 6   Simulation with precomputed transitions

Consider a homogeneous server problem with two servers holding data $(x_1, x_2) \in [0,1] \times [0,1]$. Two independent data streams arrive, one at each server. A controller has to decide to which server to route. The modeling equation for the stream shall be

$$\mathrm{d}x = b(x,u)\mathrm{d}t + \sigma(x)\mathrm{d}w, \quad u \in \{1,2\} \tag{23}$$

with

$$b(x,1) = \begin{pmatrix} 1 \\ -1 \end{pmatrix} \quad b(x,2) = \begin{pmatrix} -1 \\ 1 \end{pmatrix} \quad \sigma = \begin{pmatrix} 1 & 0 \\ 0 & 1 \end{pmatrix} \tag{24}$$

The boundaries at $x_1 = 0$ and $x_2 = 0$ are reflecting. The exceeding data on either server $x_1, x_2 > 1$ is rejected from the system and penalized with $g(x_1, 1) = g(1, x_2) = 10$, $g = 0$ otherwise. The objective of the control policy shall be to minimize

$$\mathbb{E} \int_0^\infty \mathrm{e}^{-\beta t}(x_1(t) + x_2(t) + g(x_1, x_2))dt. \tag{25}$$

The plots of the value function show, that in case of high load (i.e. $x_1, x_2$ close to 1) a maximum of cost is assumed. Therefore it is cheaper to overload a server and pay penalty than to stay close to the diagonal as is optimal in the low load case.

For simulation we used precomputed (i.e. converged heuristic) transition probabilities to test the multi-grid performance. The discount $\beta$ was set to .7. The multi-grid algorithm reduces the error in each iteration by a factor 0.21, using 5 grid levels and a $V$-cycle and two smoothing iterations on the coarsest grid. For comparison, the iteration on the finest grid converges with a reduction factor 0.63.

## 7   Discussion

We have given a condition for sampling controlled diffusion processes such that the value functions will converge while the discretization tends to zero. Rigorous numerical methods can now be applied to reinforcement learning algorithms in continuous-time, continuous-state as is demonstrated with a multi-grid algorithm for the HJB-equation. Ongoing work is directed towards adaptive grid refinement algorithms and application to systems that include hysteresis.

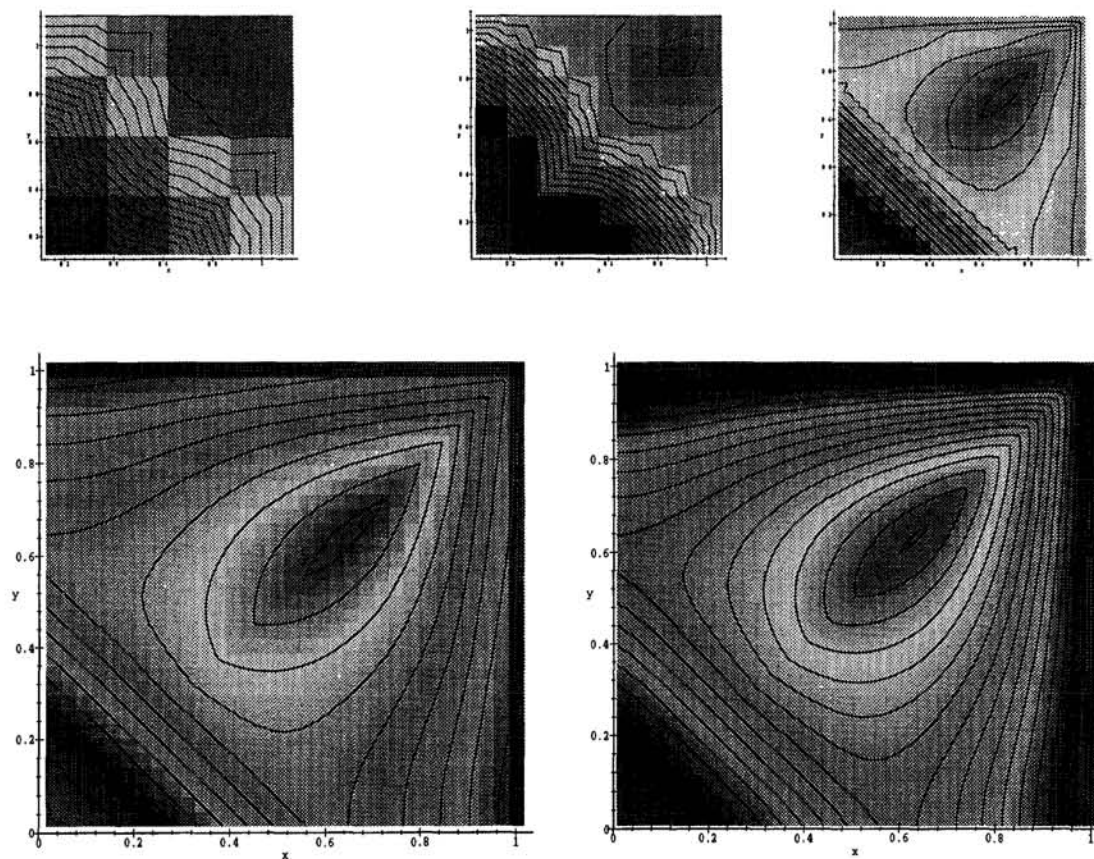

Figure 2: Contour plots of the predicted reward in a homogeneous server problem with nonlinear costs are shown on different grid levels. On the coarsest $4 \times 4$ grid a sampling rate of one second is used with 9-point-star transition matrices. At the finest grid ($64 \times 64$) a sampling rate of $\frac{1}{4}$ second is used with observation on 81-point-stars. Inside the egg-shaped area the value function assumes its maximum.

# References

[1] A. Barto, S. Bradtke, S. Singh. *Learning to Act using Real-Time Dynamic Programming*, AI Journal on Computational Theories of Interaction and Agency, 1993.

[2] M. Bloß and R. Hoppe. *Numerical Computation of the Value Function of Optimally Controlled Stochastic Switching Processes by Multi-Grid Techniques*, Numer Funct Anal And Optim 10(3+4), 275-304, 1989.

[3] S. Pareigis. *Lernen der Lösung der Bellman-Gleichung durch Beobachtung von kontinuierlichen Prozessen*, PhD Thesis, 1996.

[4] W. Hackbusch. *Multi-Grid Methods and Applications*, Springer-Verlag, 1985.

[5] H. Kushner and P. Dupuis. *Numerical Methods for Stochastic Control Problems in Continuous Time*, Springer-Verlag, 1992.
